# Local Gaussian Process Regression
# for Real Time Online Model Learning and Control

**Duy Nguyen-Tuong    Jan Peters    Matthias Seeger**
Max Planck Institute for Biological Cybernetics
Spemannstraße 38, 72076 Tübingen, Germany
`{duy,jan.peters,matthias.seeger}@tuebingen.mpg.de`

## Abstract

Learning in real-time applications, e.g., online approximation of the inverse dynamics model for model-based robot control, requires fast online regression techniques. Inspired by local learning, we propose a method to speed up standard Gaussian process regression (GPR) with local GP models (LGP). The training data is partitioned in local regions, for each an individual GP model is trained. The prediction for a query point is performed by weighted estimation using nearby local models. Unlike other GP approximations, such as mixtures of experts, we use a distance based measure for partitioning of the data and weighted prediction. The proposed method achieves online learning and prediction in real-time. Comparisons with other non-parametric regression methods show that LGP has higher accuracy than LWPR and close to the performance of standard GPR and $\nu$-SVR.

## 1  Introduction

Precise models of technical systems can be crucial in technical applications. Especially in robot tracking control, only a well-estimated inverse dynamics model can allow both high accuracy and compliant control. For complex robots such as humanoids or light-weight arms, it is often hard to model the system sufficiently well and, thus, modern regression methods offer a viable alternative [7,8]. For most real-time applications, online model learning poses a difficult regression problem due to three constraints, i.e., firstly, the learning and prediction process should be very fast (e.g., learning needs to take place at a speed of 20-200Hz and prediction at 200Hz to a 1000Hz). Secondly, the learning system needs to be capable at dealing with large amounts of data (i.e., with data arriving at 200Hz, less than ten minutes of runtime will result in more than a million data points). And, thirdly, the data arrives as a continuous stream, thus, the model has to be continuously adapted to new training examples over time.

These problems have been addressed by real-time learning methods such as locally weighted projection regression (LWPR) [7,8]. Here, the true function is approximated with local linear functions covering the relevant state-space and online learning became computationally feasible due to low computational demands of the local projection regression which can be performed in real-time. The major drawback of LWPR is the required manual tuning of many highly data-dependent metaparameters [15]. Furthermore, for complex data, large numbers of linear models are necessary in order to achieve a competitive approximation.

A powerful alternative for accurate function approximation in high-dimensional space is Gaussian process regression (GPR) [1]. Since the hyperparameters of a GP model can be adjusted by maximizing the marginal likelihood, GPR requires little effort and is easy and flexible to use. However, the main limitation of GPR is that the computational complexity scales cubically with the training examples $n$. This drawback prevents GPR from applications which need large amounts of training data and require fast computation, e.g., online learning of inverse dynamics model for model-based

robot control. Many attempts have been made to alleviate this problem, for example, (i) sparse Gaussian process (SGP) [2], and (ii) mixture of experts (ME) [3, 4]. In SGP, the training data is approximated by a smaller set of so-called inducing inputs [2, 5]. Here, the difficulty is to choose an appropriate set of inducing inputs, essentially replacing the full data set [2]. In contrast to SGP, ME divide the input space in smaller subspaces by a gating network, within which a Gaussian process expert, i.e., Gaussian local model, is trained [4, 6]. The computational cost is then significantly reduced due to much smaller number of training examples within a local model. The ME performance depends largely on the way of partitioning the training data and the choice of an optimal number of local models for a particular data set [4].

In this paper, we combine the basic idea behind both approaches, i.e., LWPR and GPR, attempting to get as close as possible to the speed of local learning while having a comparable accuracy to Gaussian process regression. This results in an approach inspired by [6, 8] using many local GPs in order to obtain a significant reduction of the computational cost during both prediction and learning step allowing the application to online learning. For partitioning the training data, we use a distance based measure, where the corresponding hyperparameters are optimized by maximizing the marginal likelihood.

The remainder of the paper is organized as follows: first, we give a short review of standard GPR in Section 2. Subsequently, we describe our local Gaussian process models (LGP) approach in Section 3 and discuss how it inherits the advantages of both GPR and LWPR. Furthermore, the learning accuracy and performance of our LGP approach will be compared with other important standard methods in Section 4, e.g., LWPR [8], standard GPR [1], sparse online Gaussian process regression (OGP) [5] and $\nu$-support vector regression ($\nu$-SVR) [11], respectively. Finally, our LGP method is evaluated for an online learning of the inverse dynamics models of real robots for accurate tracking control in Section 5. Here, the online learning is demonstrated by rank-one update of the local GP models [9]. The tracking task is performed in real-time using model-based control [10]. To our best knowledge, it is the first time that GPR is successfully used for high-speed online model learning in real time control on a physical robot. We present the results on a version of the Barrett WAM showing that with the online learned model using LGP the tracking accuracy is superior compared to state-of-the art model-based methods [10] while remaining fully compliant.

## 2 Regression with standard GPR

Given a set of $n$ training data points $\{\mathbf{x}_i, y_i\}_{i=1}^n$, we would like to learn a function $f(\mathbf{x}_i)$ transforming the input vector $\mathbf{x}_i$ into the target value $y_i$ given by $y_i = f(\mathbf{x}_i) + \epsilon_i$, where $\epsilon_i$ is Gaussian noise with zero mean and variance $\sigma_n^2$ [1]. As a result, the observed targets can also be described by $\mathbf{y} \sim \mathcal{N}\left(\mathbf{0}, \mathbf{K}(\mathbf{X}, \mathbf{X}) + \sigma_n^2 \mathbf{I}\right)$, where $\mathbf{K}(\mathbf{X}, \mathbf{X})$ denotes the covariance matrix. As covariance function, a Gaussian kernel is frequently used [1]

$$k\left(\mathbf{x}_p, \mathbf{x}_q\right) = \sigma_s^2 \exp\left(-\frac{1}{2}(\mathbf{x}_p - \mathbf{x}_q)^T \mathbf{W}(\mathbf{x}_p - \mathbf{x}_q)\right), \tag{1}$$

where $\sigma_s^2$ denotes the signal variance and $\mathbf{W}$ are the widths of the Gaussian kernel. The joint distribution of the observed target values and predicted value for a query point $\mathbf{x}_*$ is given by

$$\begin{bmatrix} \mathbf{y} \\ f(\mathbf{x}_*) \end{bmatrix} \sim \mathcal{N}\left(\mathbf{0}, \begin{bmatrix} \mathbf{K}(\mathbf{X}, \mathbf{X}) + \sigma_n^2 \mathbf{I} & \mathbf{k}(\mathbf{X}, \mathbf{x}_*) \\ \mathbf{k}(\mathbf{x}_*, \mathbf{X}) & k(\mathbf{x}_*, \mathbf{x}_*) \end{bmatrix}\right). \tag{2}$$

The conditional distribution yields the predicted mean value $f(\mathbf{x}_*)$ with the corresponding variance $V(\mathbf{x}_*)$ [1]

$$\begin{aligned} f(\mathbf{x}_*) &= \mathbf{k}_*^T \left(\mathbf{K} + \sigma_n^2 \mathbf{I}\right)^{-1} \mathbf{y} = \mathbf{k}_*^T \boldsymbol{\alpha}, \\ V(\mathbf{x}_*) &= k(\mathbf{x}_*, \mathbf{x}_*) - \mathbf{k}_*^T \left(\mathbf{K} + \sigma_n^2 \mathbf{I}\right)^{-1} \mathbf{k}_*, \end{aligned} \tag{3}$$

with $\mathbf{k}_* = \mathbf{k}(\mathbf{X}, \mathbf{x}_*)$, $\mathbf{K} = \mathbf{K}(\mathbf{X}, \mathbf{X})$ and $\boldsymbol{\alpha}$ denotes the so-called prediction vector. The hyperparameters of a Gaussian process with Gaussian kernel are $\boldsymbol{\theta} = [\sigma_n^2, \sigma_f^2, \mathbf{W}]$ and their optimal value for a particular data set can be derived by maximizing the log marginal likelihood using common optimization procedures, e.g., Quasi-Newton methods [1].

**Input:** new data point $\{\mathbf{x}, y\}$.
**for** $k = 1$ **to** number of local models **do**
    Compute distance to the $k$-th local model:
        $w_k = \exp(-0.5(\mathbf{x} - \mathbf{c}_k)^T \mathbf{W}(\mathbf{x} - \mathbf{c}_k))$
**end for**
Take the nearest local model:
    $v = \max(w_k)$
**if** $v > w_{gen}$ **then**
    Insert $\{\mathbf{x}, y\}$ to nearest local model:
        $\mathbf{X}_{\text{new}} = [\mathbf{X}, \mathbf{x}]$
        $\mathbf{y}_{\text{new}} = [\mathbf{y}, y]$
    Update corresponding center:
        $\mathbf{c}_{new} = \text{mean}(\mathbf{X}_{\text{new}})$
    Compute inverse covariance matrix and prediction vector of local model:
        $\mathbf{K}_{\text{new}} = \mathbf{K}(\mathbf{X}_{\text{new}}, \mathbf{X}_{\text{new}})$
        $\boldsymbol{\alpha}_{\text{new}} = (\mathbf{K}_{\text{new}} + \sigma^2 \mathbf{I})^{-1} \mathbf{y}_{\text{new}}$
**else**
    Create new model:
        $\mathbf{c}_{k+1} = \mathbf{x}$,
        $\mathbf{X}_{k+1} = [\mathbf{x}]$, $\mathbf{y}_{k+1} = [y]$
    Initialization new inverse covariance matrix and new prediction vector.
**end if**

**Algorithm 1:** Partitioning of training data and model learning.

**Input:** query data point $\mathbf{x}$, $M$ .
Determine $M$ local models next to $\mathbf{x}$.
**for** $k = 1$ **to** $M$ **do**
    Compute distance to the $k$-th local model:
        $w_k = \exp(-0.5(\mathbf{x} - \mathbf{c}_k)^T \mathbf{W}(\mathbf{x} - \mathbf{c}_k))$
    Compute local mean using the $k$-th local model:
        $\bar{y}_k = \mathbf{k}_k^T \boldsymbol{\alpha}_k$
**end for**
Compute weighted prediction using $M$ local models:
    $\hat{y} = \sum_{k=1}^{M} w_k \bar{y}_k / \sum_{k=1}^{M} w_k$ .

**Algorithm 2:** Prediction for a query point.

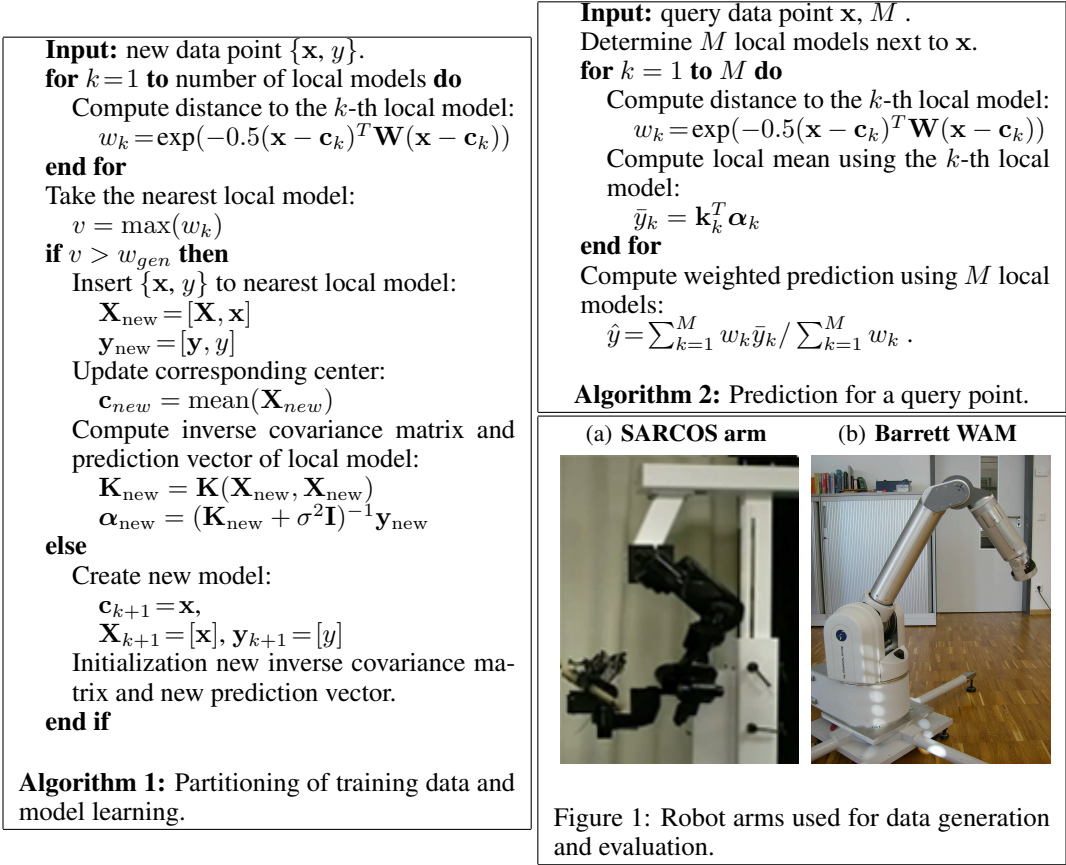

(a) **SARCOS arm**      (b) **Barrett WAM**

Figure 1: Robot arms used for data generation and evaluation.

# 3 Approximation using Local GP Models

The major limitation of GPR is the expensive computation of the inverse matrix $(\mathbf{K} + \sigma_n^2 \mathbf{I})^{-1}$ which yields a cost of $\mathcal{O}(n^3)$. Reducing this computational cost, we cluster the training data in local regions and, subsequently, train the corresponding GP models on these local clusters. The mean prediction for a query point is then made by weighted prediction using the nearby local models in the neighborhood. Thus, the algorithm consists out of two stages: (i) localization of data, i.e., allocation of new input points and learning of corresponding local models, (ii) prediction for a query point.

## 3.1 Partitioning and Training of Local Models

Clustering input data is efficiently performed by considering a distance measure of the input point $\mathbf{x}$ to the centers of all local models. The distance measure $w_k$ is given by the kernel used to learn the local GP models, e.g., Gaussian kernel

$$w_k = \exp\left(-\frac{1}{2}\left(\mathbf{x} - \mathbf{c}_k\right)^T \mathbf{W}\left(\mathbf{x} - \mathbf{c}_k\right)\right), \tag{4}$$

where $\mathbf{c}_k$ denotes the center of the $k$-th local model and $\mathbf{W}$ a diagonal matrix represented the kernel width. It should be noted, that we use the *same* kernel width for computing $w_k$ as well as for training of *all* local GP models as given in Section 2. The kernel width $\mathbf{W}$ is obtained by maximizing the log likelihood on a subset of the whole training data points. For doing so, we subsample the training data and, subsequently, perform an optimization procedure.

During the localization process, a new model with center $\mathbf{c}_{k+1}$ is created, if all distance measures $w_k$ fall below a limit value $w_{gen}$. The new data point $\mathbf{x}$ is then set as new center $\mathbf{c}_{k+1}$. Thus, the number of local models is allowed to increase as the trajectories become more complex. Otherwise, if a new point is assigned to a particular $k$-th model, the center $\mathbf{c}_k$ is updated as mean of corresponding local

data points. With the new assigned input point, the inverse covariance matrix of the corresponding local model can be updated. The localization procedure is summarized in Algorithm 1.

The main computational cost of this algorithm is $\mathcal{O}(N^3)$ for inverting the local covariance matrix, where $N$ presents the number of data points in a local model. Furthermore, we can control the complexity by limiting the number of data points in a local model. Since the number of local data points increases continuously over time, we can adhere to comply with this limit by deleting old data point as new ones are included. Insertion and deletion of data points can be decided by evaluating the information gain of the operation. The cost for inverting the local covariance matrix can be further reduced, as we need only to update the full inverse matrix once it is computed. The update can be efficiently performed in a stable manner using rank-one update [9] which has a complexity of $\mathcal{O}(N^2)$.

### 3.2 Prediction using Local Models

The prediction for a mean value $\hat{y}$ is performed using weighted averaging over $M$ local predictions $\bar{y}_k$ for a query point $\mathbf{x}$ [8]. The weighted prediction $\hat{y}$ is then given by $\hat{y} = \mathbb{E}\{\bar{y}_k|\mathbf{x}\} = \sum_{k=1}^{M} \bar{y}_k p(k|\mathbf{x})$. According to the Bayesian theorem, the probability of the model $k$ given $\mathbf{x}$ can be expressed as $p(k|\mathbf{x}) = p(k,\mathbf{x})/\sum_{k=1}^{M} p(k,\mathbf{x}) = w_k/\sum_{k=1}^{M} w_k$. Hence, we have

$$\hat{y} = \frac{\sum_{k=1}^{M} w_k \bar{y}_k}{\sum_{k=1}^{M} w_k} \ . \tag{5}$$

The probability $p(k|\mathbf{x})$ can be interpreted as a normalized distance of the query point $\mathbf{x}$ to the local model $k$ where the measure metric $w_k$ is used as given in Equation (4). Thus, each local prediction $\bar{y}_k$, determined using Equation (3), is additionally weighted by the distance $w_k$ between the corresponding center $\mathbf{c}_k$ and the query point $\mathbf{x}$. The search for $M$ local models can be quickly done by evaluating the distances between the query point $\mathbf{x}$ and all model centers $\mathbf{c}_k$. The prediction procedure is summarized in Algorithm 2.

## 4 Learning Inverse Dynamics

We have evaluated our algorithm using high-dimensional robot data taken from real robots, e.g., the 7 degree-of-freedom (DoF) anthropomorphic SARCOS master arm and 7-DoF Barrett whole arm manipulator shown in Figure 1, as well as a physically realistic SL simulation [12]. We compare the learning performance of LGP with the state-of-the-art in non-parametric regression, e.g., LWPR, $\nu$-SVR, OGP and standard GPR in the context of approximating inverse robot dynamics. For evaluating $\nu$-SVR and GPR, we have employed the libraries [13] and [14].

### 4.1 Dynamics Learning Accuracy Comparison

For the comparison of the accuracy of our method in the setting of learning inverse dynamics, we use three data sets, (i) SL simulation data (SARCOS model) as described in [15] (14094 training points, 5560 test points), (ii) data from the SARCOS master arm (13622 training points, 5500 test points) [8], (iii) a data set generated from our Barrett arm (13572 training points, 5000 test points). Given samples $\mathbf{x} = [\mathbf{q}, \dot{\mathbf{q}}, \ddot{\mathbf{q}}]$ as input, where $\mathbf{q}, \dot{\mathbf{q}}, \ddot{\mathbf{q}}$ denote the joint angles, velocity and acceleration, and using the corresponding joint torques $\mathbf{y} = [\mathbf{u}]$ as targets, we have a proper regression problem. For the considered 7 degrees of freedom robot arms, we, thus, have data with 21 input dimensions (for each joint, we have an angle, a velocity and an acceleration) and 7 targets (a torque for each joint). We learn the robot dynamics model in this 21-dim space for each DoF separately employing LWPR, $\nu$-SVR, GPR, OGP and LGP, respectively.

Partitioning of the training examples for LGP can be performed either in the same input space (where the model is learned) or in another space which has to be physically consistent with the approximated function. In the following, we localize the data depending on the position of the robot. Thus, the partitioning of training data is performed in a 7-dim space (7 joint angles). After determining $w_k$ for all $k$ local models in the partitioning space, the input point will be assigned to the *nearest* local model, i.e., the local model with the maximal value of distance measure $w_k$.

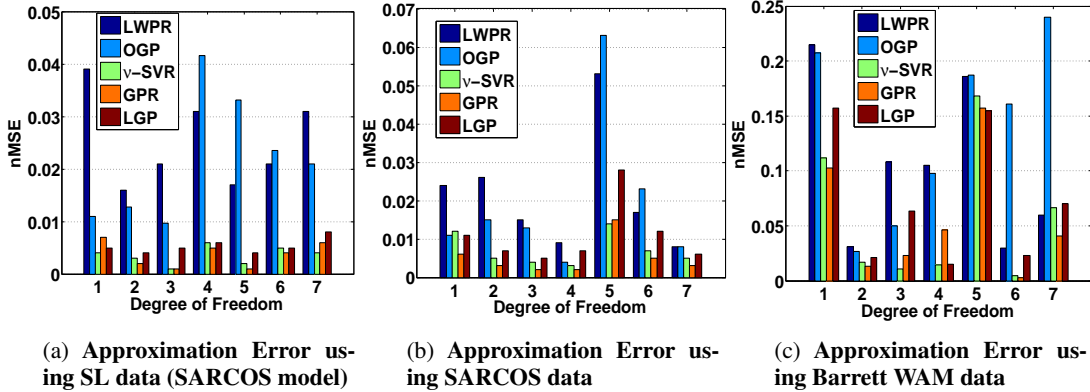

(a) **Approximation Error using SL data (SARCOS model)**

(b) **Approximation Error using SARCOS data**

(c) **Approximation Error using Barrett WAM data**

Figure 2: Approximation error as nMSE for each DoF. The error is computed after prediction on the test sets with simulated data from SL Sarcos-model, real robot data from Barrett and SARCOS master arm, respectively. In most cases, LGP outperforms LWPR and OGP in learning accuracy while being competitive to $\nu$-SVR and standard GPR. It should be noted that the nMSE depends on the target variances. Due to smaller variances in the Barrett data, the corresponding nMSE has also a larger scale compared to SARCOS.

Figure 2 shows the normalized mean squared error (nMSE) of the evaluation on the test set for each of the three evaluated scenarios, i.e., the simulated SARCOS arm in (a), the real SARCOS arm in (b) and the Barrett arm in (c). Here, the normalized mean squared error is defined as nMSE = Mean squared error/Variance of target. During the prediction on the test set using LGP, we take the most activated local models, i.e., the ones which are next to the query point.

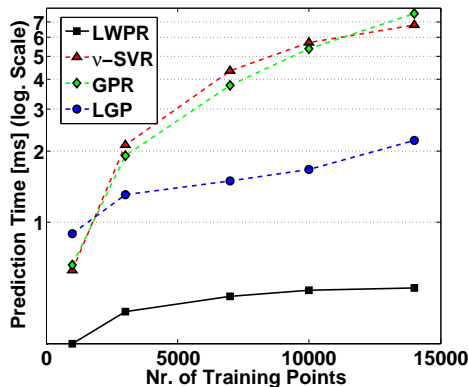

Figure 3: Average time in millisecond needed for prediction of 1 query point. The computation time is plotted logarithmic in respect of the number of training examples. The time as stated above is the required time for prediction of all 7 DoF. Here, LWPR presents the fastest method due to simple regression models. Compared to global regression methods such as standard GPR and $\nu$-SVR, local GP makes significant improvement in term of prediction time.

It should be noted that the choice of the limit value $w_{\mathrm{gen}}$ during the partitioning step is crucial for the performance of LGP and, unfortunately, is an open parameter. If $w_{\mathrm{gen}}$ is too small, a lot of local models will be generated with small number of training points. It turns out that these small local models do not perform well in generalization for unknown data. If $w_{\mathrm{gen}}$ is large, the local models become also large which increase the computational complexity. Here, the training data are clustered in about 30 local regions ensuring that each local model has a sufficient amount of data points for high accuracy (in practice, roughly a hundred data points for each local model suffice) while having sufficiently few that the solution remains feasible in real-time (on our current hardware, a Core Duo at 2GHz, that means less than 1000 data points). On average, each local model has approximately 500 training examples. This small number of training inputs enables a fast training for each local model, i.e., the matrix inversion. For estimating the hyperparameters using likelihood optimization, we subsample the training data which results in a subset of about 1000 data points.

Considering the approximation error on the test set shown in Figure 2(a-c), it can be seen that LGP generalizes well using only few local models for prediction. In all cases, LGP outperforms LWPR and OGP while being close in learning accuracy to global methods such as GPR and $\nu$-SVR. The mean-prediction for GPR is determined according to Equation (3) where we precomputed

the prediction vector $\boldsymbol{\alpha}$ from training data. When a query point appears, the kernel vector $\mathbf{k}_*^T$ is evaluated for this particular point. The operation of mean-prediction has then the order of $\mathcal{O}(n)$ for standard GPR (similarly, for $\nu$-SVR) and $\mathcal{O}(NM)$ for LGP, where $n$ denotes the total number of training points, $M$ number of local models and $N$ number of data points in a local model.

## 4.2 Comparison of Computation Speed for Prediction

Beside the reduction of training time (i.e., matrix inversion), the prediction time is also reduced significantly compared to GPR and $\nu$-SVR due to the fact that only a small amount of local models in the vicinity of the query point are needed during prediction for LGP. Thus, the prediction time can be controlled by the number of local models. A large number of local models may provide a smooth prediction but on the other hand increases the time complexity.

The comparison of prediction speed is shown in Figure 3. Here, we train LWPR, $\nu$-SVR, GPR and LGP on 5 different data sets with increasing training examples (1065, 3726, 7452, 10646 and 14904 data points, respectively). Subsequently, using the trained models we compute the average time needed to make a prediction for a query point for all 7 DoF. For LGP, we take a limited number of local models in the vicinity for prediction, e.g., $M = 3$. Since our control system requires a minimal prediction rate at 100 Hz (10 ms) in order to ensure system stability, data sets with more than 15000 points are not applicable for standard GPR or $\nu$-SVR due to high computation demands for prediction.

The results show that the computation time requirements of $\nu$-SVR and GPR rises very fast with the size of training data set as expected. LWPR remains the best method in terms of computational complexity only increasing at a very low speed. However, as shown in Figure 3, the cost for LGP is significantly lower than the one $\nu$-SVR and GPR and increases at a much lower rate. In practice, we can also curb the computation demands of single models by deleting old data points, if a new ones are assigned to the model. As approach to deleting and inserting data points, we can use the information gain of the corresponding local model as a principled measure. It can be seen from the results that LGP represents a compromise between learning accuracy and computational complexity. For large data sets (e.g., more than 5000 training examples), LGP reduces the prediction cost considerably while keeping a good learning performance.

## 5 Application in Model-based Robot Control

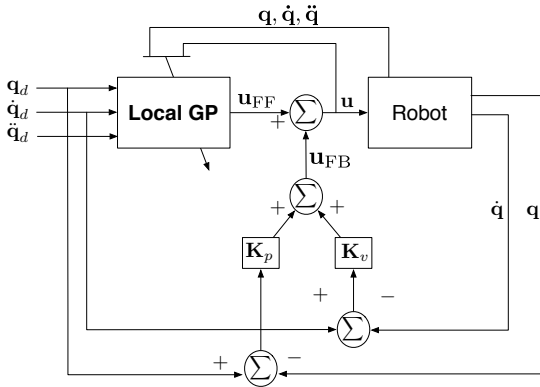

Figure 4: Schematic showing model-based robot control. The learned dynamics model can be updated online using LGP.

In this section, first, we use the inverse dynamics models learned in Section 4.1 for a model-based tracking control task [10] in the setting shown in Figure 4. Here, the learned model of the robot is applied for an online prediction of the feedforward torques $\mathbf{u}_{FF}$ given the desired trajectory $[\mathbf{q}_d, \dot{\mathbf{q}}_d, \ddot{\mathbf{q}}_d]$. Subsequently, the model approximated by LGP is used for an online learning performance. Demonstrating the online learning, the local GP models are adapted in real-time using rank-one update.

As shown in Figure 4, the controller command $\mathbf{u}$ consists of the feedforward part $\mathbf{u}_{FF}$ and the feedback part $\mathbf{u}_{FB} = \mathbf{K}_p\mathbf{e} + \mathbf{K}_v\dot{\mathbf{e}}$, where $\mathbf{e} = \mathbf{q}_d - \mathbf{q}$ denotes the tracking error and $\mathbf{K}_p, \mathbf{K}_v$ position-gain and velocity-gain, respectively. During the control experiment we set the gains to very low values taking the aim of compliant control into account. As a result, the learned model has a stronger effect on computing the predicted torque $\mathbf{u}_{FF}$ and, hence, a better learning performance of each method results in a lower tracking error.

For comparison with the learned models, we also compute the feedforward torque using rigid-body (RB) formulation which is a common approach in robot control [10]. The control task is performed

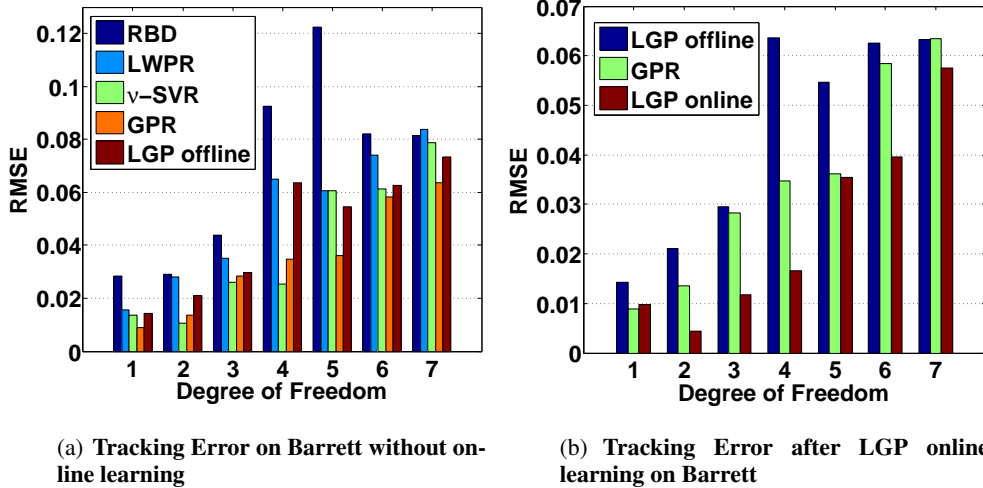

(a) **Tracking Error on Barrett without on-line learning**

(b) **Tracking Error after LGP online learning on Barrett**

Figure 5: (a) Tracking error as RMSE on test trajectory for each DoF with Barrett WAM. (b) Tracking error after online learning with LGP. The model uncertainty is reduced with online learning using LGP. With online learning, LGP is able to outperform offline learned models using standard GPR for test trajectories.

in real-time on the Barrett WAM, as shown in Figure 1. As desired trajectory, we generate a test trajectory which is similar to the one used for learning the inverse dynamics models in Section 4.1. Figure 5 (a) shows the tracking errors on test trajectory for 7 DoFs, where the error is computed as root mean squared error (RMSE). Here, LGP provides a competitive control performance compared to GPR while being superior to LWPR and the state-of-the art rigid-body model. It can be seen that for several DoFs the tracking errors are large, for example 5., 6. and 7. DoF. The reason is that for these DoFs the unknown nonlinearities are time-dependent, e.g., gear drive for 7. DoF, which can not be approximated well using just one offline learned model. Since it is not possible to learn the complete state space using a single data set, online learning is necessary.

### 5.1 Online Learning of Inverse Dynamics Models with LGP

The ability of online adaptation of the learned inverse dynamics models with LGP is shown by the rank-one update of the local models which has a complexity of $\mathcal{O}(n^2)$ [9]. Since the number of training examples in each local model is limited (500 points in average), the update procedure is fast enough for real-time application. For online learning the models are updated as shown in Figure 4.

For doing so, we regularly sample the joint torques $\mathbf{u}$ and the corresponding robot trajectories $[\mathbf{q}, \dot{\mathbf{q}}, \ddot{\mathbf{q}}]$ online. For the time being, as a new point is inserted we randomly delete another data point from the local model if the maximal number of data point is reached. The process of insertion and deletion of data points can be further improved by considering the information gain (and information lost) of the operation. Figure 5 (b) shows the tracking error after online learning with LGP. It can be seen that the errors for each DoF are significantly reduced with online LGP compared to the ones with offline learned models. With online learning, LGP is also able to outperform standard GPR.

## 6   Conclusion

We combine with LGP the fast computation of local regression with more accurate regression methods while having little tuning efforts. LGP achieves higher learning accuracy compared to locally linear methods such as LWPR while having less computational cost compared to GPR and $\nu$-SVR. The reducing cost allows LGP for model online learning which is necessary in oder to generalize the model for all trajectories. Model-based tracking control using online learned model achieves superior control performance compared to the state-of-the-art method as well as offline learned model for unknown trajectories.

# References

[1] C. E. Rasmussen and C. K. Williams, *Gaussian Processes for Machine Learning*. Massachusetts Institute of Technology: MIT-Press, 2006.

[2] J. Q. Candela and C. E. Rasmussen, "A unifying view of sparse approximate gaussian process regression," *Journal of Machine Learning Research*, 2005.

[3] V. Treps, "Mixtures of gaussian process," *Advances in Neural Information Processing Systems*, 2001.

[4] C. E. Rasmussen and Z. Ghahramani, "Infinite mixtures of gaussian process experts," *Advances in Neural Information Processing Systems*, 2002.

[5] L. Csato and M. Opper, "Sparse online gaussian processes," *Neural Computation*, 2002.

[6] E. Snelson and Z. Ghahramani, "Local and global sparse gaussian process approximations," *Artificial Intelligence and Statistics*, 2007.

[7] S. Schaal, C. G. Atkeson, and S. Vijayakumar, "Scalable techniques from nonparameteric statistics for real-time robot learning," *Applied Intelligence*, pp. 49–60, 2002.

[8] S. Vijayakumar, A. D'Souza, and S. Schaal, "Incremental online learning in high dimensions," *Neural Computation*, 2005.

[9] M. Seeger, "Low rank update for the cholesky decomposition," Tech. Rep., 2007. [Online]. Available: http://www.kyb.tuebingen.mpg.de/bs/people/seeger/

[10] J. J. Craig, *Introduction to Robotics: Mechanics and Control*, 3rd ed. Prentice Hall, 2004.

[11] B. Schölkopf and A. Smola, *Learning with Kernels: Support Vector Machines, Regularization, Optimization and Beyond*. Cambridge, MA: MIT-Press, 2002.

[12] S. Schaal, "The SL simulation and real-time control software package," Tech. Rep., 2006. [Online]. Available: http://www-clmc.usc.edu/publications/S/schaal-TRSL.pdf

[13] C.-C. Chang and C.-J. Lin, *LIBSVM: a library for support vector machines*, 2001, http://www.csie.ntu.edu.tw/ cjlin/libsvm.

[14] M. Seeger, *LHOTSE: Toolbox for Adaptive Statistical Model*, 2007, http://www.kyb.tuebingen.mpg.de/bs/people/seeger/lhotse/.

[15] D. Nguyen-Tuong, J. Peters, and M. Seeger, "Computed torque control with nonparametric regression models," *Proceedings of the 2008 American Control Conference (ACC 2008)*, 2008.

